# AdaBoost is Consistent

**Peter L. Bartlett**
Department of Statistics and Computer Science Division
University of California, Berkeley
bartlett@stat.berkeley.edu

**Mikhail Traskin**
Department of Statistics
University of California, Berkeley
mtraskin@stat.berkeley.edu

## Abstract

The risk, or probability of error, of the classifier produced by the AdaBoost algorithm is investigated. In particular, we consider the stopping strategy to be used in AdaBoost to achieve universal consistency. We show that provided AdaBoost is stopped after $n^\nu$ iterations—for sample size $n$ and $\nu < 1$—the sequence of risks of the classifiers it produces approaches the Bayes risk if Bayes risk $L^* > 0$.

## 1 Introduction

Boosting algorithms are an important recent development in classification. These algorithms belong to a group of voting methods, for example [1, 2, 3], that produce a classifier as a linear combination of *base* or *weak* classifiers. While empirical studies show that boosting is one of the best off the shelf classification algorithms (see [3]) theoretical results don't give a complete explanation of their effectiveness.

Breiman [4] showed that under some assumptions on the underlying distribution "population boosting" converges to the Bayes risk as the number of iterations goes to infinity. Since the population version assumes infinite sample size, this does not imply a similar result for AdaBoost, especially given results of Jiang [5], that there are examples when AdaBoost has prediction error asymptotically suboptimal at $t = \infty$ ($t$ is the number of iterations).

Several authors have shown that *modified* versions of AdaBoost are consistent. These modifications include restricting the $l_1$-norm of the combined classifier [6, 7] and restricting the step size of the algorithm [8]. Jiang [9] analyses the unmodified boosting algorithm and proves a process consistency property, under certain assumptions. Process consistency means that there exists a sequence $(t_n)$ such that if AdaBoost with sample size $n$ is stopped after $t_n$ iterations, its risk approaches the Bayes risk. However Jiang also imposes strong conditions on the underlying distribution: the distribution of $X$ (the predictor) has to be absolutely continuous with respect to Lebesgue measure and the function $F_B(X) = \frac{1}{2} \ln \frac{\mathbf{P}(Y=1|X)}{\mathbf{P}(Y=-1|X)}$ has to be continuous on $\mathcal{X}$. Also Jiang's proof is not constructive and does not give any hint on when the algorithm should be stopped. Bickel, Ritov and Zakai in [10] prove a consistency result for AdaBoost, under the assumption that the probability distribution is such that the steps taken by the algorithm are not too large. We would like to obtain a simple stopping rule that guarantees consistency and doesn't require any modification to the algorithm.

This paper provides a constructive answer to all of the mentioned issues:

1. We consider AdaBoost (not a modification).

2. We provide a simple stopping rule: the number of iterations $t$ is a fixed function of the sample size $n$.

3. We assume only that the class of base classifiers has finite VC-dimension, and that the span of this class is sufficiently rich. Both assumptions are clearly necessary.

## 2 Setup and notation

Here we describe the AdaBoost procedure formulated as a coordinate descent algorithm and introduce definitions and notation. We consider a binary classification problem. We are given $\mathcal{X}$, the measurable (feature) space, and $\mathcal{Y} = \{-1, 1\}$, set of (binary) labels. We are given a sample $S_n = \{(X_i, Y_i)\}_{i=1}^n$ of i.i.d. observations distributed as the random variable $(X, Y) \sim \mathcal{P}$, where $\mathcal{P}$ is an unknown distribution. Our goal is to construct a classifier $g_n : \mathcal{X} \to \mathcal{Y}$ based on this sample. The quality of the classifier $g_n$ is given by the misclassification probability

$$L(g_n) = \mathbf{P}(g_n(X) \neq Y | S_n).$$

Of course we want this probability to be as small as possible and close to the Bayes risk

$$L^* = \inf_g L(g) = \mathrm{E}(\min\{\eta(X), 1 - \eta(X)\}),$$

where the infimum is taken over all possible (measurable) classifiers and $\eta(\cdot)$ is a conditional probability

$$\eta(x) = \mathbf{P}(Y = 1 | X = x).$$

The infimum above is achieved by the Bayes classifier $g^*(x) = g(2\eta(x) - 1)$, where

$$g(x) = \left\{ \begin{array}{rcl} 1 & , & x > 0, \\ -1 & , & x \leq 0. \end{array} \right.$$

We are going to produce a classifier as a linear combination of *base* classifiers in $\mathcal{H} = \{h | h : \mathcal{X} \to \mathcal{Y}\}$. We shall assume that class $\mathcal{H}$ has a finite VC (Vapnik-Chervonenkis) dimension $d_{VC}(\mathcal{H}) = \max\{|S| : S \subseteq \mathcal{X}, |\mathcal{H}_{|S}| = 2^{|S|}\}$.

Define

$$R_n(f) = \frac{1}{n} \sum_{i=1}^n e^{-Y_i f(X_i)} \qquad \text{and} \qquad R(f) = \mathrm{E} e^{-Y f(X)}.$$

Then the boosting procedure can be described as follows.

1. Set $f_0 \equiv 0$, choose number of iterations $t$.
2. For $k = 1, \ldots, t$ set
$$f_k = f_{k-1} + \alpha_{k-1} h_{k-1},$$
where the following holds
$$R_n(f_k) = \inf_{h \in \mathcal{H}, \alpha \in \mathbb{R}} R_n(f_{k-1} + \alpha h) \tag{1}$$
We call $\alpha_i$ the step size of the algorithm at step $i$.
3. Output $g \circ f_t$ as a final classifier.

We shall also use the convex hull of $\mathcal{H}$ scaled by $\lambda \geq 0$,

$$\mathcal{F}_\lambda = \left\{ f \,\middle|\, f = \sum_{i=1}^n \lambda_i h_i, n \in \mathbb{N} \cup \{0\}, \lambda_i \geq 0, \sum_{i=1}^n \lambda_i = \lambda, h_i \in \mathcal{H} \right\}$$

as well as the set of $k$-combinations, $k \in \mathbb{N}$, of functions in $\mathcal{H}$

$$\mathcal{F}^k = \left\{ f \,\middle|\, f = \sum_{i=1}^k \lambda_i h_i, \lambda_i \in \mathbb{R}, h_i \in \mathcal{H} \right\}.$$

We shall also need to define the $l_*$-norm: for any $f \in \mathcal{F}$

$$\|f\|_* = \inf\{\sum |\alpha_i|, f = \sum \alpha_i h_i, h_i \in \mathcal{H}\}.$$

Define the squashing function $\pi_l(\cdot)$ to be

$$\pi_l(x) = \left\{ \begin{array}{rcl} l & , & x > l, \\ x & , & x \in [-l, l], \\ -l & , & x < -l. \end{array} \right.$$

Then the set of truncated functions is

$$\pi_l \circ \mathcal{F} = \left\{ \tilde{f} \mid \tilde{f} = \pi_l(f), f \in \mathcal{F} \right\}.$$

The set of classifiers based on class $\mathcal{F}$ is denoted by

$$g \circ \mathcal{F} = \{ \tilde{f} \mid \tilde{f} = g(f), f \in \mathcal{F} \}.$$

Define the derivative of an arbitrary function $Q(\cdot)$ in the direction of $h$ as

$$Q'(f; h) = \left. \frac{\partial Q(f + \lambda h)}{\partial \lambda} \right|_{\lambda = 0}.$$

The second derivative $Q''(f; h)$ is defined similarly.

## 3   Consistency of boosting procedure

We shall need the following assumption.

**Assumption 1** *Let the distribution $\mathcal{P}$ and class $\mathcal{H}$ be such that*

$$\lim_{\lambda \to \infty} \inf_{f \in \mathcal{F}_\lambda} R(f) = R^*,$$

*where $R^* = \inf R(f)$ over all measurable functions.*

For many classes $\mathcal{H}$, the above assumption is satisfied for all possible distributions $\mathcal{P}$. See [6, Lemma 1] for sufficient conditions for Assumption 1. As an example of such a class, we can take a class of indicators of all rectangles or indicators of half-spaces defined by hyperplanes or binary trees with the number of terminal nodes equal to $d+1$ (we consider trees with terminal nodes formed by successive univariate splits), where $d$ is the dimensionality of $\mathcal{X}$ (see [4]).

We begin with a simple lemma (see [1, Theorem 8] or [11, Theorem 6.1]):

**Lemma 1** *For any $t \in \mathbb{N}$ if $d_{VC}(\mathcal{H}) \geq 2$ the following holds:*

$$d_P(\mathcal{F}^t) \leq 2(t+1)(d_{VC}(\mathcal{H}) + 1) \log_2[2(t+1)/\ln 2],$$

*where $d_P(\mathcal{F}^t)$ is the pseudodimension of class $\mathcal{F}^t$.*

The proof of AdaBoost consistency is based on the following result, which builds on the result by Koltchinskii and Panchenko [12] and resembles [6, Lemma 2].

**Lemma 2** *For a continuous function $\varphi$ define the Lipschitz constant*

$$L_{\varphi,\lambda} = \inf\{L \mid L > 0, |\varphi(x) - \varphi(y)| \leq L|x - y|, -\lambda \leq x, y \leq \lambda\}$$

*and maximum absolute value of $\varphi(\cdot)$ when argument is in $[-\lambda, \lambda]$*

$$M_{\varphi,\lambda} = \max_{x \in [-\lambda, \lambda]} \varphi(x).$$

*Then for functions*

$$R_\varphi(f) = \mathrm{E}\varphi(Yf(X)) \qquad and \qquad R_{\varphi,n}(f) = \frac{1}{n} \sum_{i=1}^n \varphi(Y_i f(X_i)),$$

$V = d_{VC}(\mathcal{H})$, $c = 24 \int_0^1 \sqrt{\ln \frac{8e}{\epsilon^2}} \, d\epsilon$ *and any $n$, $\lambda > 0$ and $t > 0$,*

$$\mathrm{E} \sup_{f \in \pi_\lambda \circ \mathcal{F}^t} |R_\varphi(f) - R_{\varphi,n}(f)| \leq c\lambda L_{\varphi,\lambda} \sqrt{\frac{(V+1)(t+1) \log_2[2(t+1)/\ln 2]}{n}} \qquad (2)$$

*and*

$$\mathrm{E} \sup_{f \in \mathcal{F}_\lambda} |R_\varphi(f) - R_{\varphi,n}(f)| \leq 4\lambda L_{\varphi,\lambda} \sqrt{\frac{2V \ln(4n+2)}{n}}. \qquad (3)$$

*Also, for any $\delta > 0$, with probability at least $1 - \delta$,*

$$\sup_{f \in \pi_\lambda \circ \mathcal{F}^t} |R_\varphi(f) - R_{\varphi,n}(f)| \leq c\lambda L_{\varphi,\lambda} \sqrt{\frac{(V+1)(t+1)\log_2[2(t+1)/\ln 2]}{n}}$$
$$+ M_{\varphi,\lambda} \sqrt{\frac{\ln(1/\delta)}{2n}} \qquad (4)$$

*and*

$$\sup_{f \in \mathcal{F}_\lambda} |R_\varphi(f) - R_{\varphi,n}(f)| \leq 4\lambda L_{\varphi,\lambda} \sqrt{\frac{2V\ln(4n+2)}{n}} + M_{\varphi,\lambda} \sqrt{\frac{\ln(1/\delta)}{2n}}. \qquad (5)$$

*Proof.* Equations (3) and (5) constitute [6, Lemma 2]. The proof of equations (2) and (4) is similar. We begin with symmetrization to get

$$\mathrm{E} \sup_{f \in \pi_\lambda \circ \mathcal{F}^t} |R_\varphi(f) - R_{\varphi,n}(f)| \leq 2\mathrm{E} \sup_{f \in \pi_\lambda \circ \mathcal{F}^t} \left| \frac{1}{n} \sum_{i=1}^{n} \sigma_i(\varphi(-Y_i f(X_i)) - \varphi(0)) \right|,$$

where $\sigma_i$ are i.i.d. with $\mathbf{P}(\sigma_i = 1) = \mathbf{P}(\sigma_i = -1) = 1/2$. Then we use the "contraction principle" (see [13, Theorem 4.12, pp. 112–113]) with a function $\psi(x) = (\varphi(x) - \varphi(0))/L_{\varphi,\lambda}$ to get

$$\mathrm{E} \sup_{f \in \pi_\lambda \circ \mathcal{F}^t} |R_\varphi(f) - R_{\varphi,n}(f)| \leq 4L_{\varphi,\lambda}\mathrm{E} \sup_{f \in \pi_\lambda \circ \mathcal{F}^t} \left| \frac{1}{n} \sum_{i=1}^{n} -\sigma_i Y_i f(X_i) \right|$$
$$= 4L_{\varphi,\lambda}\mathrm{E} \sup_{f \in \pi_\lambda \circ \mathcal{F}^t} \left| \frac{1}{n} \sum_{i=1}^{n} \sigma_i f(X_i) \right|.$$

Next we proceed and find the supremum. Notice, that functions in $\pi_\lambda \circ \mathcal{F}^t$ are bounded and clipped to the absolute value equal $\lambda$, therefore we can rescale $\pi_\lambda \circ \mathcal{F}^t$ by $(2\lambda)^{-1}$ and get

$$\mathrm{E} \sup_{f \in \pi_\lambda \circ \mathcal{F}^t} \left| \frac{1}{n} \sum_{i=1}^{n} \sigma_i f(X_i) \right| = 2\lambda\mathrm{E} \sup_{f \in (2\lambda)^{-1} \circ \pi_\lambda \circ \mathcal{F}^t} \left| \frac{1}{n} \sum_{i=1}^{n} \sigma_i f(X_i) \right|.$$

Next, we are going to use Dudley's entropy integral [14] to bound the r.h.s above

$$\mathrm{E} \sup_{f \in (2\lambda)^{-1} \circ \pi_\lambda \circ \mathcal{F}^t} \left| \frac{1}{n} \sum_{i=1}^{n} \sigma_i f(X_i) \right| \leq \frac{12}{\sqrt{n}} \int_0^\infty \sqrt{\ln \mathcal{N}(\epsilon, (2\lambda)^{-1} \circ \pi_\lambda \circ \mathcal{F}^t, L_2(P_n))} d\epsilon.$$

Since for $\epsilon > 1$ the covering number $\mathcal{N}$ is 1, then upper integration limit can be taken 1, and we can use Pollard's bound [15] for $F \subseteq [0,1]^\mathcal{X}$

$$\mathcal{N}(\epsilon, F, L_2(P)) \leq 2 \left( \frac{4e}{\epsilon^2} \right)^{d_P(F)},$$

where $d_P(F)$ is a pseudodimension, and obtain for $\tilde{c} = 12 \int_0^1 \sqrt{\ln \frac{8e}{\epsilon^2}} d\epsilon$

$$\mathrm{E} \sup_{f \in (2\lambda)^{-1} \circ \pi_\lambda \circ \mathcal{F}^t} \left| \frac{1}{n} \sum_{i=1}^{n} \sigma_i f(X_i) \right| \leq \tilde{c} \sqrt{\frac{d_P((2\lambda)^{-1} \circ \pi_\lambda \circ \mathcal{F}^t)}{n}},$$

also notice that constant $\tilde{c}$ doesn't depend on $\mathcal{F}^t$ or $\lambda$. Next, since $(2\lambda)^{-1} \circ \pi_\lambda$ is a non-decreasing transform, we use inequality $d_P((2\lambda)^{-1} \circ \pi_\lambda \circ \mathcal{F}^t) \leq d_P(\mathcal{F}^t)$ (e.g. [11, Theorem 11.3])

$$\mathrm{E} \sup_{f \in (2\lambda)^{-1} \circ \pi_\lambda \circ \mathcal{F}^t} \left| \frac{1}{n} \sum_{i=1}^{n} \sigma_i f(X_i) \right| \leq c \sqrt{\frac{d_P(\mathcal{F}^t)}{n}}.$$

And then, since Lemma 1 gives an upper-bound on the pseudodimension of the class $\mathcal{F}^t$, we have

$$\mathrm{E} \sup_{f \in \pi_\lambda \circ \mathcal{F}^t} \left| \frac{1}{n} \sum_{i=1}^{n} \sigma_i f(X_i) \right| \leq c\lambda \sqrt{\frac{(V+1)(t+1)\log_2[2(t+1)/\ln 2]}{n}},$$

with constant $c$ above being independent of $\mathcal{H}$, $t$ and $\lambda$. To prove the second statement we use McDiarmid's bounded difference inequality [16, Theorem 9.2, p. 136], since $\forall i$

$$\sup_{(x_j,y_j)_{j=1}^n,(x_i',y_i')} \left| \sup_{f\in\pi_\lambda\circ\mathcal{F}^t} |R_\varphi(f) - R_{\varphi,n}(f)| - \sup_{f\in\pi_\lambda\circ\mathcal{F}^t} |R_\varphi(f) - R_{\varphi,n}'(f)| \right| \leq \frac{M_{\varphi,\lambda}}{n},$$

where $R_{\varphi,n}'(f)$ is obtained from $R_{\varphi,n}(f)$ by changing pair $(x_i, y_i)$ to $(x_i', y_i')$. This completes the proof of the lemma. ◇

Lemma 2, unlike [6, Lemma 2], allows us to choose the number of steps $t$, that describes the complexity of the linear combination of base functions in addition to the parameter $\lambda$, which governs the size of the deviations of the functions in $\mathcal{F}$, and this is essential for the proof of the consistency. It is easy to see that for AdaBoost (i.e. $\varphi(x) = e^{-x}$) we have to choose $\lambda = \kappa \ln n$ and $t = n^\nu$ with $\kappa > 0$, $\nu > 0$ and $2\kappa + \nu < 1$. So far we dealt with the statistical properties of the function we are minimizing, now we turn to the algorithmic part. We need the following simple consequence of the proof of [10, Theorem 1]

**Theorem 1** *Let function $Q(f)$ be convex in $f$. Let $Q^* = \lim_{\lambda\to\infty} \inf_{f\in\mathcal{F}_\lambda} Q(f)$. Assume that $\forall c_1, c_2$, s.t. $Q^* < c_1 < c_2 < \infty$,*

$$\begin{aligned} 0 &< \inf\{Q''(f;h) : c_1 < Q(f) < c_2, h \in \mathcal{H}\} \\ &\leq \sup\{Q''(f;h) : Q(f) < c_2, h \in \mathcal{H}\} < \infty. \end{aligned}$$

*Then for any reference function $\bar{f}$ and the sequence of functions $f_m$, produced by the boosting algorithm, the following bound holds $\forall m$ s.t. $Q(f_m) > Q(\bar{f})$.*

$$Q(f_m) \leq Q(\bar{f}) + \sqrt{\frac{8B^3 Q(f_0)(Q(f_0) - Q(\bar{f}))}{\beta^3}} \left( \ln \frac{\ell_0^2 + c_3(m+1)}{\ell_0^2} \right)^{-\frac{1}{2}}, \tag{6}$$

*where $\ell_k = \left\| \bar{f} - f_k \right\|_*$, $c_3 = 2Q(f_0)/\beta$, $\beta = \inf\{Q''(f;h) : Q(\bar{f}) < Q(f) < Q(f_0), h \in \mathcal{H}\}$, $B = \sup\{Q''(f;h) : Q(f) < Q(f_0), h \in \mathcal{H}\}$.*

*Proof.* The statement of the theorem is a version of the result implicit in the proof of [10, Theorem 1]. If for some $m$ we have $Q(f_m) \leq Q(\bar{f})$, then theorem is trivially true for all $m' \geq m$. Therefore, we are going to consider only the case when $Q(f_{m+1}) > Q(\bar{f})$. By convexity of $Q(\cdot)$

$$|Q'(f_m; f_m - \bar{f})| \geq Q(f_m) - Q(\bar{f}) = \epsilon_m. \tag{7}$$

Let $f_m - \bar{f} = \sum \tilde{\alpha}_i \tilde{h}_i$, where $\tilde{\alpha}_i$ and $\tilde{h}_i$ correspond to the best representation (with the smallest $l_*$-norm). Then from (7) and linearity of the derivative we have

$$\epsilon_m \leq \left| \sum \tilde{\alpha}_i Q'(f_m; \tilde{h}_i) \right| \leq \sup_{h\in\mathcal{H}} |Q'(f_m; h)| \sum |\tilde{\alpha}_i|,$$

therefore

$$\sup_{h\in\mathcal{H}} Q'(f_m; h) \geq \frac{\epsilon_m}{\left\| f_m - \bar{f} \right\|_*}. \tag{8}$$

Next,

$$Q(f_m + \alpha h_m) = Q(f_m) + \alpha Q'(f_m; h_m) + \frac{1}{2}\alpha^2 Q''(\tilde{f}_m; h_m),$$

where $\tilde{f}_m = f_m + \tilde{\alpha}_m h_m$, for $\tilde{\alpha}_m \in [0, \alpha_m]$, and since by assumption $\tilde{f}_m$ is on the path from $f_m$ to $f_{m+1}$ we have the following bounds

$$Q(\bar{f}) < Q(f_{m+1}) \leq Q(\tilde{f}_m) \leq Q(f_m) \leq Q(f_0),$$

then by assumption of the theorem for $\beta$, that depends on $Q(\bar{f})$, we have

$$Q(f_{m+1}) \geq Q(f_m) + \inf_{\alpha\in\mathbb{R}}(\alpha Q'(f_m; h_m) + \frac{1}{2}\alpha^2 \beta) = Q(f_m) - \frac{|Q'(f_m; h_m)|^2}{2\beta}. \tag{9}$$

On the other hand,

$$\begin{aligned} Q(f_m + \alpha_m h_m) &= \inf_{h\in\mathcal{H},\alpha\in\mathbb{R}} Q(f_m + \alpha h) \leq \inf_{h\in\mathcal{H},\alpha\in\mathbb{R}} \left( Q(f_m) + \alpha Q'(f_m; h) + \frac{1}{2}\alpha^2 B \right) \\ &= Q(f_m) - \frac{\sup_{h\in\mathcal{H}} |Q'(f_m; h)|^2}{2B}. \end{aligned} \tag{10}$$

Therefore, combining (9) and (10) , we get

$$|Q'(f_m; h_m)| \geq \sup_{h \in \mathcal{H}} |Q'(f_m; h)| \sqrt{\frac{\beta}{B}}. \tag{11}$$

Another Taylor expansion, this time around $f_{m+1}$, gives us

$$Q(f_m) = Q(f_{m+1}) + \frac{1}{2}\alpha_m^2 Q''(\tilde{\tilde{f}}_m; h_m), \tag{12}$$

where $\tilde{\tilde{f}}_m$ is some (other) function on the path from $f_m$ to $f_{m+1}$. Therefore, if $|\alpha_m| < |Q'(f_m; h_m)|/B$, then

$$Q(f_m) - Q(f_{m+1}) < \frac{|Q'(f_m; h_m)|^2}{2B},$$

but by (10)

$$Q(f_m) - Q(f_{m+1}) \geq \frac{\sup_{h \in \mathcal{H}} |Q'(f_m; h)|^2}{2B} \geq \frac{|Q'(f_m; h_m)|^2}{2B},$$

therefore we conclude, by combining (11) and (8), that

$$|\alpha_m| \geq \frac{|Q'(f_m; h_m)|}{B} \geq \frac{\sqrt{\beta} \sup_{h \in \mathcal{H}} |Q'(f_m; h)|}{B^{3/2}} \geq \frac{\epsilon_m \sqrt{\beta}}{\ell_m B^{3/2}}. \tag{13}$$

Using (12) we have

$$\sum_{i=0}^{m} \alpha_i^2 \leq \frac{2}{\beta} \sum_{i=0}^{m} (Q(f_i) - Q(f_{i+1})) \leq \frac{2}{\beta}(Q(f_0) - Q(\bar{f})). \tag{14}$$

Recall that

$$\left\| f_m - \bar{f} \right\|_* \leq \left\| f_{m-1} - \bar{f} \right\|_* + |\alpha_{m-1}| \leq \left\| f_0 - \bar{f} \right\|_* + \sum_{i=0}^{m-1} |\alpha_i|$$

$$\leq \left\| f_0 - \bar{f} \right\|_* + \sqrt{m} \left( \sum_{i=0}^{m-1} \alpha_i^2 \right)^{1/2},$$

therefore, combining with (14) and (13), since sequence $\epsilon_i$ is decreasing,

$$\frac{2}{\beta}(Q(f_0) - Q(\bar{f})) \geq \sum_{i=0}^{m} \alpha_i^2 \geq \frac{\beta}{B^3} \sum_{i=0}^{m} \frac{\epsilon_i^2}{\ell_i^2} \geq \frac{\beta}{B^3} \epsilon_m^2 \sum_{i=0}^{m} \frac{1}{\left( \ell_0 + \sqrt{i} \left( \sum_{j=0}^{i-1} \alpha_j^2 \right)^{1/2} \right)^2}$$

$$\geq \frac{\beta}{B^3} \epsilon_m^2 \sum_{i=0}^{m} \frac{1}{\left( \ell_0 + \sqrt{i} \left( \frac{2Q(f_0)}{\beta} \right)^{1/2} \right)^2}$$

$$\geq \frac{\beta}{2B^3} \epsilon_m^2 \sum_{i=0}^{m} \frac{1}{\ell_0^2 + \frac{2Q(f_0)}{\beta} i}.$$

Since

$$\sum_{i=0}^{m} \frac{1}{a + bi} \geq \int_0^{m+1} \frac{dx}{a + bx} = \frac{1}{b} \ln \frac{a + b(m+1)}{a},$$

then

$$\frac{2}{\beta}(Q(f_0) - Q(\bar{f})) \geq \frac{\beta^2}{4B^3 Q(f_0)} \epsilon_m^2 \ln \frac{\ell_0^2 + \frac{2Q(f_0)}{\beta}(m+1)}{\ell_0^2}.$$

Therefore

$$\epsilon_m \leq \sqrt{\frac{8B^3 Q(f_0)(Q(f_0) - Q(\bar{f}))}{\beta^3}} \left( \ln \frac{\ell_0^2 + \frac{2Q(f_0)}{\beta}(m+1)}{\ell_0^2} \right)^{-\frac{1}{2}},$$

and this completes the proof. ◇

The theorem above allows us to get an upper bound on the difference between the $\varphi$-risk of the function output by AdaBoost and the $\varphi$-risk of the appropriate reference function.

**Theorem 2** *Assume $R^* > 0$. Let $t_n = n^\nu$ be the number of steps we run AdaBoost, let $\lambda_n = \kappa \ln n$, with $\nu > 0$, $\kappa > 0$ and $\nu + 2\kappa < 1$. Let $\bar{f}_n$ be a minimizer of the function $R_n(\cdot)$ within $\mathcal{F}_{\lambda_n}$. Then for $n$ large enough with high probability the following holds*

$$R_n(f_{t_n}) \leq R_n(\bar{f}_n) + \frac{8}{(R^*)^{3/2}} \left( \ln \frac{\lambda_n^2 + (4/R^*)t_n}{\lambda_n^2} \right)^{-1/2}$$

*Proof.* This theorem follows directly from Theorem 1. Because in AdaBoost

$$R_n''(f;h) = \frac{1}{n} \sum_{i=1}^{n} (-Y_i h(X_i))^2 e^{-Y_i f(X_i)} = \frac{1}{n} \sum_{i=1}^{n} e^{-Y_i f(X_i)} = R(f)$$

then all the conditions in Theorem 1 are satisfied (with $Q(f)$ replaced by $R_n(f)$) and in the Equation (6) we have $B = R_n(f_0) = 1$, $\beta \geq R_n(\bar{f}_n)$, $\|f_0 - \bar{f}_n\|_* \leq \lambda_n$. Since for $t$ s.t. $R_n(f_t) \leq R_n(\bar{f}_n)$ the theorem is trivially true we only have to notice that Lemma 2 guarantees that with probability at least $1 - \delta$

$$|R(\bar{f}_n) - R_n(\bar{f}_n)| \leq 4\lambda_n L_{\varphi,\lambda_n} \sqrt{\frac{2V \ln(4n+2)}{n}} + M_{\varphi,\lambda_n} \sqrt{\frac{\ln(1/\delta)}{2n}}.$$

Thus for $n$ such that the r.h.s. of the above expression is less than $R^*/2$ we have $\beta \geq R_n(\bar{f}_n) \geq R^*/2$ and the result follows immediately from Equation (6) if we use the fact that $R_n(\bar{f}) > 0$. $\diamond$

Then, having all the ingredients at hand we can formulate the main result of the paper.

**Theorem 3** *Assume $V = d_{VC}(\mathcal{H}) < \infty$, $L^* > 0$,*

$$\lim_{\lambda \to \infty} \inf_{f \in \mathcal{F}_\lambda} R(f) = R^*,$$

$t_n \to \infty$, *and* $t_n = O(n^\nu)$ *for* $\nu < 1$. *Then AdaBoost stopped at step $t_n$ returns a sequence of classifiers almost surely satisfying $L(g(f_{t_n})) \to L^*$.*

*Proof.* For the exponential loss function $L^* > 0$ implies $R^* > 0$. Let $\lambda_n = \kappa \ln n$, $\kappa > 0$, $2\kappa + \nu < 1$. Also, let $\bar{f}$ be a minimizer of $R$ and $\bar{f}_n$ be a minimizer of $R_n$ within $\mathcal{F}_{\lambda_n}$. Then we have

$$
\begin{aligned}
R(\pi_{\lambda_n}(f_{t_n})) &\leq R_n(\pi_{\lambda_n}(f_{t_n})) + \epsilon_1 \quad \text{by Lemma 2} && (15)\\
&\leq R_n(f_{t_n}) + \epsilon_1 + \varphi(\lambda_n) \quad \text{since } \varphi(\pi_{\lambda_n}(x)) \leq \varphi(x) + \varphi(\lambda_n) \\
&\leq R_n(\bar{f}_n) + \epsilon_1 + \varphi(\lambda_n) + \epsilon_2 \quad \text{by Theorem 2} && (16)\\
&\leq R(\bar{f}) + \epsilon_1 + \varphi(\lambda_n) + \epsilon_2 + \epsilon_3 \quad \text{by Lemma 2.} && (17)
\end{aligned}
$$

Inequalities (15) and (17) hold with probability at least $1 - \delta_n$, while inequality (16) is true for sufficiently large $n$ when (17) holds. The $\epsilon$'s above are

$$\epsilon_1 = c n^\kappa \kappa \ln n \sqrt{\frac{(V+1)(n^\nu+1)\log_2[2(n^\nu+1)/\ln 2]}{n}} + n^\kappa \sqrt{\frac{\ln(1/\delta_n)}{2n}}$$

$$\epsilon_2 = \frac{8}{(R^*)^{3/2}} \left( \ln \frac{(\kappa \ln n)^2 + (4/R^*)n^\nu}{(\kappa \ln n)^2} \right)^{-1/2},$$

$$\epsilon_3 = 4 n^\kappa \kappa \ln n \sqrt{\frac{2V \ln(4n+2)}{n}} + n^\kappa \sqrt{\frac{\ln(1/\delta_n)}{2n}}$$

and $\varphi(\lambda_n) = n^{-\kappa}$. Therefore, by the choice of $\nu$ and $\kappa$ and appropriate choice of $\delta_n$, for example $\delta_n = n^{-2}$, we have $\epsilon_1 \to 0$, $\epsilon_2 \to 0$, $\epsilon_3 \to 0$ and $\varphi(\lambda_n) \to 0$. Also, $R(\bar{f}) \to R^*$ by Assumption 1. Now we appeal to the Borel-Cantelli lemma and arrive at $R(\pi_\lambda(f_{t_n})) \to R^*$ a.s. Eventually we can use [17, Theorem 3] to conclude that

$$L(g(\pi_{\lambda_n}(f_{t_n}))) \overset{a.s.}{\to} L^*.$$

But for $\lambda_n > 0$ we have $g(\pi_{\lambda_n}(f_{t_n})) = g(f_{t_n})$, therefore

$$L(g(f_{t_n})) \overset{a.s.}{\to} L^*.$$

Hence AdaBoost is consistent if stopped after $n^\nu$ steps. $\diamond$

## 4  Discussion

We showed that AdaBoost is consistent if stopped sufficiently early, after $t_n$ iterations, for $t_n = n^\nu$ with $\nu < 1$, given that Bayes risk $L^* > 0$. It is unclear whether this number can be increased. Results by Jiang [5] imply that for some $\mathcal{X}$ and function class $\mathcal{H}$ AdaBoost algorithm will achieve zero training error after $t_n$ steps, where $n^2/t_n = o(1)$. We don't know what happens in between $O(n^{1-\varepsilon})$ and $O(n^2 \ln n)$. Lessening this gap is a subject of further research.

We analyzed only AdaBoost, the boosting algorithm that uses loss function $\varphi(x) = e^{-x}$. Since the proof of Theorem 2 relies on the properties of the exponential loss, we cannot make a similar conclusion for other versions of boosting, e.g., logit boosting with $\varphi(x) = \ln(1 + e^{-x})$: in this case assumption on the second derivative holds with $R_n''(f;h) \geq R_n(f)/n$, though the resulting inequality is trivial, the factor $1/n$ precludes us from finding any useful bound. It is a subject of future work to find an analog of Theorem 2 that will handle logit loss.

#### Acknowledgments

We gratefully acknowledge the support of NSF under award DMS-0434383.

### References

[1] Yoav Freund and Robert E. Schapire. A decision-theoretic generalization of on-line learning and an application to boosting. *Journal of Computer and System Sciences*, 55(1):119–139, 1997.

[2] Leo Breiman. Bagging predictors. *Machine Learning*, 24(2):123–140, 1996.

[3] Leo Breiman. Arcing classifiers (with discussion). *The Annals of Statistics*, 26(3):801–849, 1998. (Was Department of Statistics, U.C. Berkeley Technical Report 460, 1996).

[4] Leo Breiman. Some infinite theory for predictor ensembles. Technical Report 579, Department of Statistics, University of California, Berkeley, 2000.

[5] Wenxin Jiang. On weak base hypotheses and their implications for boosting regression and classification. *The Annals of Statistics*, 30:51–73, 2002.

[6] Gábor Lugosi and Nicolas Vayatis. On the Bayes-risk consistency of regularized boosting methods. *The Annals of Statistics*, 32(1):30–55, 2004.

[7] Tong Zhang. Statistical behavior and consistency of classification methods based on convex risk minimization. *The Annals of Statistics*, 32(1):56–85, 2004.

[8] Tong Zhang and Bin Yu. Boosting with early stopping: convergence and consistency. *The Annals of Statistics*, 33:1538–1579, 2005.

[9] Wenxin Jiang. Process consistency for AdaBoost. *The Annals of Statistics*, 32(1):13–29, 2004.

[10] P. J. Bickel, Y. Ritov, and A. Zakai. Some theory for generalized boosting algorithms. *Journal of Machine Learning Research*, 7:705–732, May 2006.

[11] Martin Anthony and Peter Bartlett. *Neural network learning: theoretical foundations*. Cambridge University Press, 1999.

[12] V. Koltchinskii and D. Panchenko. Empirical margin distributions and bounding the generalization error of combined classifiers. *The Annals of Statistics*, 30:1–50, 2002.

[13] Michel Ledoux and Michel Talagrand. *Probability in Banach Spaces*. Springer-Verlag, New York, 1991.

[14] Richard M. Dudley. *Uniform central limit theorems*. Cambridge University Press, Cambridge, MA, 1999.

[15] David Pollard. *Empirical Processes: Theory and Applications*. IMS, 1990.

[16] Luc Devroye, László Györfi, and Gábor Lugosi. *A Probabilistic Theory of Pattern Recognition*. Springer, New York, 1996.

[17] Peter L. Bartlett, Michael I. Jordan, and Jon D. McAuliffe. Convexity, classification, and risk bounds. *Journal of the American Statistical Association*, 101(473):138–156, 2006.
